# Knowledge-Based Support Vector Machine Classifiers

**Glenn M. Fung, Olvi L. Mangasarian and Jude W. Shavlik**
Computer Sciences Department, University of Wisconsin
Madison, WI 53706
*gfung,olvi,shavlik@cs.wisc.edu*

## Abstract

Prior knowledge in the form of multiple polyhedral sets, each belonging to one of two categories, is introduced into a reformulation of a linear support vector machine classifier. The resulting formulation leads to a linear program that can be solved efficiently. Real world examples, from DNA sequencing and breast cancer prognosis, demonstrate the effectiveness of the proposed method. Numerical results show improvement in test set accuracy after the incorporation of prior knowledge into ordinary, data-based linear support vector machine classifiers. One experiment also shows that a linear classifier, based solely on prior knowledge, far outperforms the direct application of prior knowledge rules to classify data.

**Keywords:** *use and refinement of prior knowledge, support vector machines, linear programming*

## 1  Introduction

Support vector machines (SVMs) have played a major role in classification problems [18, 3, 11]. However unlike other classification tools such as knowledge-based neural networks [16, 17, 7], little work [15] has gone into incorporating prior knowledge into support vector machines. In this work we present a novel approach to incorporating prior knowledge in the form of polyhedral *knowledge sets* in the input space of the given data. These knowledge sets, which can be as simple as cubes, are supposed to belong to one of two categories into which all the data is divided. Thus, a single knowledge set can be interpreted as a generalization of a training example, which typically consists of a single *point* in input space. In contrast, each of our knowledge sets consists of a *region* in the same space. By using a powerful tool from mathematical programming, theorems of the alternative [9, Chapter 2], we are able to embed such prior data into a linear program that can be efficiently solved by any of the publicly available solvers.

We briefly summarize the contents of the paper now. In Section 2 we describe the linear support vector machine classifier and give a linear program for it. We then describe how prior knowledge, in the form of polyhedral knowledge sets belonging to one of two classes can be characterized. In Section 3 we incorporate these polyhedral sets into our linear programming formulation which results in our knowledge-based support vector machine (KSVM) formulation (19). This formulation is capable of generating a linear classifier based on real data and/or prior knowledge. Section 4 gives a brief summary of numerical results that compare various linear and nonlinear classifiers with and without the incorporation of prior knowledge. Section 5 concludes the paper.

We now describe our notation. All vectors will be column vectors unless transposed to a row vector by a prime $'$. The scalar (inner) product of two vectors $x$ and $y$ in the $n$-dimensional real space $R^n$ will be denoted by $x'y$. For a vector $x$ in $R^n$, the sign function $sign(x)$ is defined as $sign(x)_i = 1$ if $x_i > 0$ else $sign(x)_i = -1$ if $x_i \leq 0$, for $i = 1, \ldots, n$. For $x \in R^n$, $\|x\|_p$ denotes the $p$-norm, $p = 1, 2, \infty$. The notation $A \in R^{m \times n}$ will signify a real $m \times n$ matrix. For such a matrix, $A'$ will denote the transpose of $A$ and $A_i$ will denote the $i$-th row of $A$. A vector of ones in a real space of arbitrary dimension will be denoted by $e$. Thus for $e \in R^m$ and $y \in R^m$ the notation $e'y$ will denote the sum of the components of $y$. A vector of zeros in a real space of arbitrary dimension will be denoted by $0$. The identity matrix of arbitrary dimension will be denoted by $I$. A *separating plane*, with respect to two given point sets $\mathcal{A}$ and $\mathcal{B}$ in $R^n$, is a plane that attempts to separate $R^n$ into two halfspaces such that each open halfspace contains points mostly of $\mathcal{A}$ or $\mathcal{B}$. A *bounding plane* to the set $\mathcal{A}$ is a plane that places $\mathcal{A}$ in one of the two closed halfspaces that the plane generates. The symbol $\wedge$ will denote the logical "and". The abbreviation "s.t." stands for "such that".

## 2 Linear Support Vector Machines and Prior Knowledge

We consider the problem, depicted in Figure 1(a), of classifying $m$ points in the $n$-dimensional input space $R^n$, represented by the $m \times n$ matrix $A$, according to membership of each point $A_i$ in the class $A^+$ or $A^-$ as specified by a given $m \times m$ diagonal matrix $D$ with plus ones or minus ones along its diagonal. For this problem, the linear programming support vector machine [11, 2] with a linear kernel, which is a variant of the standard support vector machine [18, 3], is given by the following linear program with parameter $\nu > 0$:

$$\min_{(w,\gamma,y) \in R^{n+1+m}} \{\nu e'y + \|w\|_1 \mid D(Aw - e\gamma) + y \geq e, y \geq 0\}, \qquad (1)$$

where $\| \cdot \|_1$ denotes the 1-norm as defined in the Introduction, $y$ is a vector of slack variables measuring empirical error and $(w, \gamma)$ characterize a separating plane depicted in Figure 1. That this problem is indeed a linear program, can be easily seen from the equivalent formulation:

$$\min_{(w,\gamma,y,t) \in R^{n+1+m+n}} \{\nu e'y + e't \mid D(Aw - e\gamma) + y \geq e, t \geq w \geq -t, y \geq 0\}, \qquad (2)$$

where $e$ is a vector of ones of appropriate dimension. For economy of notation we shall use the first formulation (1) with the understanding that computational implementation is via (2). As depicted in Figure 1(a), $w$ is the normal to the bounding planes:

$$x'w = \gamma + 1, \ x'w = \gamma - 1, \qquad (3)$$

that bound the points belonging to the sets $A^+$ and $A^-$ respectively. The constant $\gamma$ determines their location relative to the origin. When the two classes are strictly linearly separable, that is when the error variable $y = 0$ in (1) (which is the case shown in Figure 1(a)), the plane $x'w = \gamma + 1$ bounds *all* of the class $A^+$ points, while the plane $x'w = \gamma - 1$ bounds *all* of the class $A^-$ points as follows:

$$A_i w \ \geq \ \gamma + 1, \text{ for } D_{ii} = 1, \ A_i w \ \leq \ \gamma - 1, \text{ for } D_{ii} = -1. \qquad (4)$$

Consequently, the plane:

$$x'w = \gamma, \qquad (5)$$

midway between the bounding planes (3), is a separating plane that separates points belonging to $A^+$ from those belonging to $A^-$ completely if $y = 0$, else only approximately. The 1-norm term $\|w\|_1$ in (1), which is half the reciprocal of the distance $\frac{2}{\|w\|_1}$ measured using the $\infty$-norm distance [10] between the two bounding planes of

(3) (see Figure 1(a)), maximizes this distance, often called the "margin". Maximizing the margin enhances the generalization capability of a support vector machine [18, 3]. If the classes are linearly inseparable, then the two planes bound the two classes with a "soft margin" (i.e. bound approximately with some error) determined by the nonnegative error variable $y$, that is:

$$A_i w + y_i \geq \gamma + 1, \ \text{for } D_{ii} = 1, \ A_i w - y_i \leq \gamma - 1, \ \text{for } D_{ii} = -1. \qquad (6)$$

The 1-norm of the error variable $y$ is minimized parametrically with weight $\nu$ in (1), resulting in an approximate separating plane (5) which classifies as follows:

$$x \in A^+ \ \text{if } sign(x'w - \gamma) = 1, \ x \in A^- \ \text{if } sign(x'w - \gamma) = -1. \qquad (7)$$

Suppose now that we have prior information of the following type. All points $x$ lying in the polyhedral set determined by the linear inequalities:

$$Bx \leq b, \qquad (8)$$

belong to class $A^+$. Such inequalities generalize simple box constraints such as $a \leq x \leq d$. Looking at Figure 1(a) or at the inequalities (4) we conclude that the following implication must hold:

$$Bx \leq b \implies x'w \geq \gamma + 1. \qquad (9)$$

That is, the *knowledge set* $\{x \mid Bx \leq b\}$ lies on the $A^+$ side of the bounding plane $x'w = \gamma + 1$. Later, in (19), we will accommodate the case when the implication (9) cannot be satisfied exactly by the introduction of slack error variables. For now, assuming that the implication (9) holds for a given $(w, \gamma)$, it follows that (9) is equivalent to:

$$Bx \leq b, \ x'w < \gamma + 1, \ \text{has no solution } x. \qquad (10)$$

This statement in turn is *implied* by the following statement:

$$B'u + w = 0, \ b'u + \gamma + 1 \leq 0, \ u \geq 0, \ \text{has a solution } (u, w). \qquad (11)$$

To see this simple backward implication: (10)$\Longleftarrow$(11), we suppose the contrary that there exists an $x$ satisfying (10) and obtain the contradiction $b'u > b'u$ as follows:

$$b'u \geq u'Bx = -w'x > -\gamma - 1 \geq b'u, \qquad (12)$$

where the first inequality follows by premultiplying $Bx \leq b$ by $u \geq 0$. In fact, under the natural assumption that the prior knowledge set $\{x \mid Bx \leq b\}$ is nonempty, the forward implication: (10)$\Longrightarrow$(11) is also true, as a direct consequence of the nonhomogeneous Farkas theorem of the alternative [9, Theorem 2.4.8]. We state this equivalence as the following key proposition to our knowledge-based approach.

**Proposition 2.1 Knowledge Set Classification.** *Let the set $\{x \mid Bx \leq b\}$ be nonempty. Then for a given $(w, \gamma)$, the implication (9) is equivalent to the statement (11). In other words, the set $\{x \mid Bx \leq b\}$ lies in the halfspace $\{x \mid w'x \geq \gamma + 1\}$ if and only if there exists $u$ such that $B'u + w = 0$, $b'u + \gamma + 1 \leq 0$ and $u \geq 0$.*

**Proof** We establish the equivalence of (9) and (11) by showing the equivalence (10) and (11). By the nonhomogeneous Farkas theorem [9, Theorem 2.4.8] we have that (10) is equivalent to either:

$$B'u + w = 0, \ b'u + \gamma + 1 \leq 0, \ u \geq 0, \ \text{having solution } (u, w), \qquad (13)$$

or $\qquad\qquad B'u = 0, \ b'u < 0, \ u \geq 0, \ \text{having solution } u. \qquad (14)$

However, the second alternative (14) contradicts the nonemptiness of the knowledge-set $\{x \mid Bx \leq b\}$, because for $x$ in this set and $u$ solving (14) gives the contradiction:

$$0 \geq u'(Bx - b) = x'B'u - b'u = -b'u > 0. \qquad (15)$$

Hence (14) is ruled out and we have that (10) is equivalent to (13) which is (11). $\square$

This proposition will play a key role in incorporating knowledge sets, such as $\{x \mid Bx \leq b\}$, into one of two categories in a support vector classifier formulation as demonstrated in the next section.

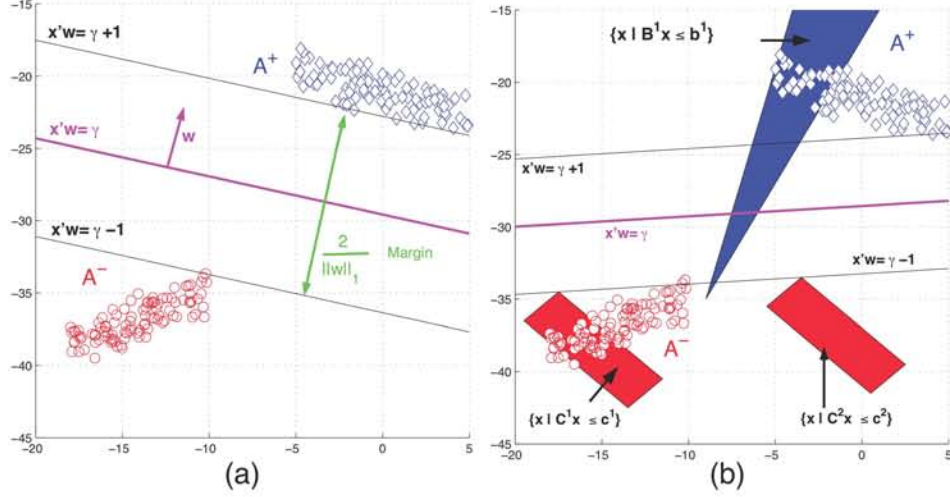

Figure 1: **(a): A linear SVM separation for 200 points in** $R^2$ **using the linear programming formulation (1). (b): A linear SVM separation for the same 200 points in** $R^2$ **as those in Figure 1(a) but using the linear programming formulation (19) which incorporates three knowledge sets:** $\{x \mid B^1 x \leq b^1\}$ **into the halfspace of** $A^+$, **and** $\{x \mid C^1 x \leq c^1\}$, $\{x \mid C^2 x \leq c^2\}$ **into the halfspace of** $A^-$, **as depicted above. Note the substantial difference between the linear classifiers** $x'w = \gamma$ **of both figures.**

## 3   Knowledge-Based SVM Classification

We describe now how to incorporate prior knowledge in the form of polyhedral sets into our linear programming SVM classifier formulation (1).

We assume that we are given the following *knowledge sets*:

$$
\begin{aligned}
&k \text{ sets belonging to } A^+ : \{x \mid B^i x \leq b^i\}, \ i = 1, \ldots, k \\
&\ell \text{ sets belonging to } A^- : \{x \mid C^i x \leq c^i\}, \ i = 1, \ldots, \ell
\end{aligned}
\tag{16}
$$

It follows by Proposition 2.1 that, relative to the bounding planes (3):

$$
\begin{aligned}
&\text{There exist } u^i, \ i = 1, \ldots, k, \ v^j, \ j = 1, \ldots, \ell, \text{ such that:} \\
&B^{i'} u^i + w = 0, \ b^{i'} u^i + \gamma + 1 \leq 0, \ u^i \geq 0, \ i = 1, \ldots, k \\
&C^{j'} v^j - w = 0, \ c^{j'} v^j - \gamma + 1 \leq 0, \ v^j \geq 0, \ j = 1, \ldots, \ell
\end{aligned}
\tag{17}
$$

We now incorporate the knowledge sets (16) into the SVM linear programming formulation (1) classifier, by adding the conditions (17) as constraints to it as follows:

$$
\begin{aligned}
\min_{w, \gamma, (y, u^i, v^j) \geq 0} \quad & \nu e' y + \|w\|_1 \\
\text{s.t.} \quad D(Aw - e\gamma) + y \ &\geq \ e \\
B^{i'} u^i + w \ &= \ 0 \\
b^{i'} u^i + \gamma + 1 \ &\leq \ 0, \ i = 1, \ldots, k \\
C^{j'} v^j - w \ &= \ 0 \\
c^{j'} v^j - \gamma + 1 \ &\leq \ 0, \ j = 1, \ldots, \ell
\end{aligned}
\tag{18}
$$

This linear programming formulation will ensure that each of the knowledge sets $\{x \mid B^i x \leq b^i\}$, $i = 1, \ldots, k$ and $\{x \mid C^i x \leq c^i\}$, $i = 1, \ldots, \ell$ lie on the appropriate side of the bounding planes (3). However, there is no guarantee that such bounding planes exist that will precisely separate these two classes of knowledge sets, just as there is no *a priori* guarantee that the original points belonging to the sets $A^+$ and $A^-$ are linearly separable. We therefore add error variables

$r^i, \rho^i, i = 1, \ldots, k, s^j, \sigma^j, j = 1, \ldots, \ell$, just like the slack error variable $y$ of the SVM formulation (1), and attempt to drive these error variables to zero by modifying our last formulation above as follows:

$$\min_{w, \gamma, (y, u^i, r^i, \rho^i, v^j, s^j, \sigma^j) \geq 0} \nu e' y + \mu(\sum_{i=1}^{k}(r^i + \rho^i) \quad + \quad \sum_{j=1}^{\ell}(s^j + \sigma^j)) + \|w\|_1$$

$$\begin{aligned} \text{s.t.} \quad D(Aw - e\gamma) + y &\geq e \\ -r^i \leq B^{i'}u^i + w &\leq r^i \\ b^{i'}u^i + \gamma + 1 &\leq \rho^i, \ i = 1, \ldots, k \\ -s^j \leq C^{j'}v^j - w &\leq s^j \\ c^{j'}v^j - \gamma + 1 &\leq \sigma^j, \ j = 1, \ldots, \ell \end{aligned} \qquad (19)$$

This is our final knowledge-based linear programming formulation which incorporates the knowledge sets (16) into the linear classifier with weight $\mu$, while the (empirical) error term $e'y$ is given weight $\nu$. As usual, the value of these two parameters, $\nu, \mu$, are chosen by means of a tuning set extracted from the training set. If we set $\mu = 0$ then the linear program (19) degenerates to (1), the linear program associated with an ordinary linear SVM. However, if set $\nu = 0$, then the linear program (19) generates a linear SVM that is strictly based on knowledge sets, but not on any specific training data. This might be a useful paradigm for situations where training datasets are not easily available, but expert knowledge, such as doctors' experience in diagnosing certain diseases, is readily available. This will be demonstrated in the breast cancer dataset of Section 4.

Note that the 1-norm term $\|w\|_1$ can be replaced by one half the 2-norm squared, $\frac{1}{2}\|w\|_2^2$, which is the usual margin maximization term for ordinary support vector machine classifiers [18, 3]. However, this changes the linear program (19) to a quadratic program which typically takes longer time to solve.

For standard SVMs, support vectors consist of all data points which are the complement of the data points that can be dropped from the problem without changing the separating plane (5) [18, 11]. Thus for our knowledge-based linear programming formulation (19), support vectors correspond to data points (rows of the matrix $A$) for which the Lagrange multipliers are nonzero, because solving (19) with these data points only will give the same answer as solving (19) with the entire matrix $A$.

The concept of support vectors has to be modified as follows for our knowledge sets. Since each knowledge set in (16) is represented by a matrix $B^i$ or $C^j$, each row of these matrices can be thought of as characterizing a boundary plane of the knowledge set. In our formulation (19) above, such rows are wiped out if the corresponding components of the variables $u^i$ or $v^j$ are zero at an optimal solution. We call the *complement* of these components of the the knowledge sets (16), *support constraints*. Deleting constraints (rows of $B^i$ or $C^j$), for which the corresponding components of $u^i$ or $v^j$ are zero, will not alter the solution of the knowledge-based linear program (19). This in fact is corroborated by numerical tests that were carried out. Deletion of non-support constraints can be considered a *refinement of prior knowledge* [17]. Another type of of refinement of prior knowledge may occur when the separating plane $x'w = \gamma$ intersects one of the knowledge sets. In such a case the plane $x'w = \gamma$ can be added as an inequality to the knowledge set it intersects. This is illustrated in the following example.

We demonstrate the geometry of incorporating knowledge sets by considering a synthetic example in $R^2$ with $m = 200$ points, 100 of which are in $A^+$ and the other 100 in $A^-$. Figure 1(a) depicts ordinary linear separation using the linear SVM formulation (1). We now incorporate three knowledge sets into the the problem:

$\{x \mid B^1 x \leq b^1\}$ belonging to $A^+$ and $\{x \mid C^1 x \leq c^1\}$ and $\{x \mid C^2 x \leq c^2\}$ belonging to $A^-$, and solve our linear program (19) with $\mu = 100$ and $\nu = 1$. We depict the new linear separation in Figure 1(b) and note the substantial change generated in the linear separation by the incorporation of these three knowledge sets. Also note that since the plane $x'w = \gamma$ intersects the knowledge set $\{x \mid B^1 x \leq b^1\}$, this knowledge set can be refined to the following $\{x \mid B^1 x \leq b^1, \; w'x \geq \gamma\}$.

# 4    Numerical Testing

Numerical tests, which are described in detail in [6], were carried out on the DNA promoter recognition dataset [17] and the Wisconsin prognostic breast cancer dataset WPBC (ftp://ftp.cs.wisc.edu/math-prog/cpo-dataset/machine-learn/cancer/WPBC/). We briefly summarize these results here.

Our first dataset, the promoter recognition dataset, is from the the domain of DNA sequence analysis. A *promoter*, which is a short DNA sequence that precedes a gene sequence, is to be distinguished from a *nonpromoter*. Promoters are important in identifying starting locations of genes in long uncharacterized sequences of DNA. The prior knowledge for this dataset, which consists of a set of 14 prior rules, matches none of the examples of the training set. Hence these rules by themselves cannot serve as a classifier. However, they do capture significant information about promoters and it is known that incorporating them into a classifier results in a more accurate classifier [17]. These 14 prior rules were converted in a straightforward manner [6] into 64 knowledge sets. Following the methodology used in prior work [17], we tested our algorithm on this dataset together with the knowledge sets, using a "leave-one-out" cross validation methodology in which the entire training set of 106 elements is repeatedly divided into a training set of size 105 and a test set of size 1. The values of $\nu$ and $\mu$ associated with both KSVM and SVM$_1$ [2] where obtained by a tuning procedure which consisted of varying them on a square grid: $\{2^{-6}, 2^{-5}, \ldots, 2^6\} \times \{2^{-6}, 2^{-5}, \ldots, 2^6\}$. After expressing the prior knowledge in the form of polyhedral sets and applying KSVM, we obtained 5 errors out of 106 (5/106). KSVM gave a much better performance than five other different methods that do not use prior knowledge: Standard 1-norm support vector machine [2] (9/106), Quinlan's decision tree builder [13] (19/106), PEBLS Nearest algorithm [4] with $k = 3$ (13/106), an empirical method suggested by a biologist based on a collection of "filters" to be used for promoter recognition known as O'Neill's Method [12] (12/106), neural networks with a simple connected layer of hidden units trained using back-propagation [14] (8/106). Except for KSVM and SVM$_1$, all of these results are taken from an earlier report [17]. KSVM was also compared with [16] where a hybrid learning system maps problem specific prior knowledge, represented in propositional logic into neural networks and then, refines this reformulated knowledge using back propagation. This method is known as Knowledge Based Artificial Neural Networks (**KBANN**). KBANN was the only approach that performed slightly better than our algorithm and obtained 4 misclassifications compared to our 5. However, it is important to note that our classifier is a much simpler linear classifier, $sign(x'w - \gamma)$, while the neural network classifier of KBANN is a considerably more complex nonlinear classifier. Furthermore, we note that KSVM is simpler to implement than KBANN and requires merely a commonly available linear programming solver. In addition, KSVM which is a linear support vector machine classifier, improves by 44.4% the error of an ordinary linear 1-norm SVM classifier that does not utilize prior knowledge sets.

The second dataset used in our numerical tests was the Wisconsin breast cancer prognosis dataset WPBC using a 60-month cutoff for predicting recurrence or nonrecurrence of the disease [2]. The prior knowledge utilized in this experiment consisted of the prognosis rules used by doctors [8] which depended on two features from the dataset: tumor size (T)(feature 31), that is the diameter of the excised tumor in

centimeters and lymph node status (L) which refers to the number of metastasized axillary lymph nodes (feature 32). The rules are:

$$(L \geq 5) \wedge (T \geq 4) \Longrightarrow RECUR \quad \text{and} \quad (L = 0) \wedge (T \leq 1.9) \Longrightarrow NONRECUR$$

It is important to note that the rules described above can be applied directly to classify *only* 32 of the given 110 given points of the training dataset and correctly classify 22 of these 32 points. The remaining 78 points are not classifiable by the above rules. Hence, if the rules are applied as a classifier by themselves the classification accuracy would be 20%. As such, these rules are not very useful by themselves and doctors use them in conjunction with other rules [8]. However, using our approach the rules were converted to linear inequalities and used in our KSVM algorithm without any use of the data, i.e. $\nu = 0$ in the linear program (19). The resulting linear classifier in the 2-dimensional space of L(ymph) and T(umor) achieved 66.4% accuracy. The ten-fold, cross-validated test set correctness achieved by standard SVM using all the data is 66.2% [2]. This result is remarkable because our knowledge-based formulation can be applied to problems where training data may not be available whereas expert knowledge may be readily available in the form of knowledge sets. This fact makes this method considerably different from previous hybrid methods like KBANN where training examples are needed in order to refine prior knowledge. If training data are added to this knowledge-based formulation, no noticeable improvement is obtained.

## 5  Conclusion & Future Directions

We have proposed an efficient procedure for incorporating prior knowledge in the form of knowledge sets into a linear support vector machine classifier either in combination with a given dataset or based solely on the knowledge sets. This novel and promising approach of handling prior knowledge is worthy of further study, especially ways to handle and simplify the combinatorial nature of incorporating prior knowledge into linear inequalities. A class of possible future applications might be to problems where training data may not be easily available whereas expert knowledge may be readily available in the form of knowledge sets. This would correspond to solving our knowledge based linear program (19) with $\nu = 0$. A typical example of this type was breast cancer prognosis [8] where knowledge sets by themselves generated a linear classifier as good as any classifier based on data points. This is a new way of incorporating prior knowledge into powerful support vector machine classifiers. Also, the concept of support constraints as discussed at the end of Section 3, warrants further study that may lead to a systematic simplification of prior knowledge sets. Other avenues of research include, knowledge sets characterized by nonpolyhedral convex sets as well as nonlinear kernels [18, 11] which are capable of handling more complex classification problems, as well as the incorporation of prior knowledge into multiple instance learning [1, 5] which might lead to improved classifiers in that field.

## Acknowledgments

Research in this UW Data Mining Institute Report 01-09, November 2001, was supported by NSF Grants CCR-9729842, IRI-9502990 and CDA-9623632, by AFOSR Grant F49620-00-1-0085, by NLM Grant 1 R01 LM07050-01, and by Microsoft.

## References

[1] P. Auer. On learning from multi-instance examples: Empirical evaluation of a theoretical approach. pages 21–29, 1987.

[2] P. S. Bradley and O. L. Mangasarian. Feature selection via concave minimization and support vector machines. In J. Shavlik, editor, *Machine Learning Proceedings of the Fifteenth International Conference(ICML '98)*, pages 82–90, San

Francisco, California, 1998. Morgan Kaufmann. ftp://ftp.cs.wisc.edu/math-prog/tech-reports/98-03.ps.

[3] V. Cherkassky and F. Mulier. *Learning from Data - Concepts, Theory and Methods*. John Wiley & Sons, New York, 1998.

[4] S. Cost and S. Salzberg. A weighted nearest neighbor algorithm for learning with symbolic features. *Machine Learning*, 10:57–58, 1993.

[5] T. G. Dietterich, R. H. Lathrop, and T. Lozano-Perez. Solving the multiple-instance problem with axis-parallel rectangles. *Artificial Intelligence*, 89:31–71, 1998.

[6] G. Fung, O. L. Mangasarian, and J. Shavlik. Knowledge-based support vector machine classifiers. Technical Report 01-09, Data Mining Institute, Computer Sciences Department, University of Wisconsin, Madison, Wisconsin, November 2001. ftp://ftp.cs.wisc.edu/pub/dmi/tech-reports/01-09.ps.

[7] F. Girosi and N. Chan. Prior knowledge and the creation of "virtual" examples for RBF networks. In *Neural networks for signal processing, Proceedings of the 1995 IEEE-SP Workshop*, pages 201–210, New York, 1995. IEEE Signal Processing Society.

[8] Y.-J. Lee, O. L. Mangasarian, and W. H. Wolberg. Survival-time classification of breast cancer patients. Technical Report 01-03, Data Mining Institute, Computer Sciences Department, University of Wisconsin, Madison, Wisconsin, March 2001. *Computational Optimization and Applications*, to appear. ftp://ftp.cs.wisc.edu/pub/dmi/tech-reports/01-03.ps.

[9] O. L. Mangasarian. *Nonlinear Programming*. SIAM, Philadelphia, PA, 1994.

[10] O. L. Mangasarian. Arbitrary-norm separating plane. *Operations Research Letters*, 24:15–23, 1999. ftp://ftp.cs.wisc.edu/math-prog/tech-reports/97-07r.ps.

[11] O. L. Mangasarian. Generalized support vector machines. In A. Smola, P. Bartlett, B. Schölkopf, and D. Schuurmans, editors, *Advances in Large Margin Classifiers*, pages 135–146, Cambridge, MA, 2000. MIT Press. ftp://ftp.cs.wisc.edu/math-prog/tech-reports/98-14.ps.

[12] M. C. O'Neill. Escherichia coli promoters: I. concensus as it relates to spacing class, specificity, repeat substructure, and three dimensional organization. *Journal of Biological Chemistry*, 264:5522–5530, 1989.

[13] J. R. Quinlan. *Induction of Decision Trees*, volume 1. 1986.

[14] D. E. Rumelhart, G. E. Hinton, and R. J. Williams. Learning internal representations by error propagation. In D. E. Rumelhart and J. L. McClelland, editors, *Parallel Distributed Processing*, pages 318–362, Cambridge, Massachusetts, 1986. MIT Press.

[15] B. Schölkopf, P. Simard, A. Smola, and V. Vapnik. Prior knowledge in support vector kernels. In M. Jordan, M. Kearns, and S. Solla, editors, *Advances in Neural Information Processing Systems 10*, pages 640 – 646, Cambridge, MA, 1998. MIT Press.

[16] G. G. Towell and J. W. Shavlik. Knowledge-based artificial neural networks. *Artificial Intelligence*, 70:119–165, 1994.

[17] G. G. Towell, J. W. Shavlik, and M. Noordewier. Refinement of approximate domain theories by knowledge-based artificial neural networks. In *Proceedings of the Eighth National Conference on Artificial Intelligence (AAAI-90)*, pages 861–866, 1990.

[18] V. N. Vapnik. *The Nature of Statistical Learning Theory*. Springer, New York, second edition, 2000.
